# Optimal Unsupervised Motor Learning Predicts the Internal Representation of Barn Owl Head Movements

Terence D. Sanger
Jet Propulsion Laboratory
MS 303-310
4800 Oak Grove Drive
Pasadena, CA 91109

## Abstract

(Masino and Knudsen 1990) showed some remarkable results which suggest that head motion in the barn owl is controlled by distinct circuits coding for the horizontal and vertical components of movement. This implies the existence of a set of orthogonal internal coordinates that are related to meaningful coordinates of the external world. No coherent computational theory has yet been proposed to explain this finding. I have proposed a simple model which provides a framework for a theory of low-level motor learning. I show that the theory predicts the observed microstimulation results in the barn owl. The model rests on the concept of "Optimal Unsupervised Motor Learning", which provides a set of criteria that predict optimal internal representations. I describe two iterative Neural Network algorithms which find the optimal solution and demonstrate possible mechanisms for the development of internal representations in animals.

## 1   INTRODUCTION

In the sensory domain, many algorithms for unsupervised learning have been proposed. These algorithms learn depending on statistical properties of the input data, and often can be used to find useful "intermediate" sensory representations

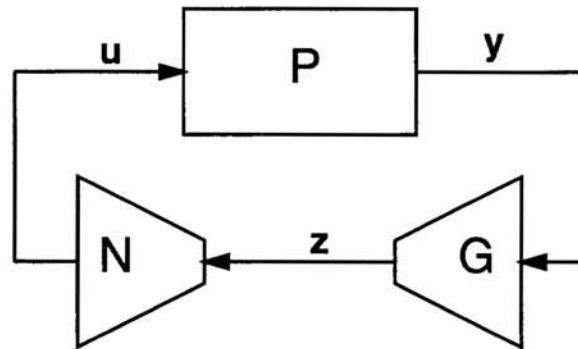

Figure 1: Structure of Optimal Unsupervised Motor Learning. $z$ is a reduced-order internal representation between sensory data $y$ and motor commands $u$. $P$ is the plant and $G$ and $N$ are adaptive sensory and motor networks. A desired value of $z$ produces a motor command $u = Nz$ resulting in a new intermediate value $\hat{z} = GPNz$.

by extracting important features from the environment (Kohonen 1982, Sanger 1989, Linsker 1989, Becker 1992, for example). An extension of these ideas to the domain of motor control has been proposed in (Sanger 1993). This work defined the concept of "Optimal Unsupervised Motor Learning" as a method for determining optimal internal representations for movement. These representations are intended to model the important controllable components of the sensory environment, and neural networks are capable of learning the computations necessary to gain control of these components.

In order to use this theory as a model for biological systems, we need methods to infer the form of biological internal representations so that these representations can be compared to those predicted by the theory. Discrepancies between the predictions and results may be due either to incorrect assumptions in the model, or to constraints on biological systems which prevent them from achieving optimality. In either case, such discrepancies can lead to improvements in the model and are thus important for our understanding of the computations involved. On the other hand, if the model succeeds in making qualitative predictions of biological responses, then we can claim that the biological system possesses the optimality properties of the model, although it is unlikely to perform its computations in exactly the same manner.

## 2   BARN OWL EXPERIMENTS

A relevant set of experiments was performed by (Masino and Knudsen 1990) in the barn owl. These experiments involved microstimulation of sites in the optic tectum responsible for head movement. By studying the responses to stimulation at different sites separated by short or long time intervals, it was possible to infer the existence of distinct "channels" for head movement which could be made refractory by prior stimulation. These channels were oriented in the horizontal and vertical directions in external coordinates, despite the fact that the neck musculature of the barn owl is sufficiently complex that such orientations appear unrelated to any set

of natural motor coordinates. This result raises two related questions. First, why are the two channels orthogonal with respect to external Cartesian coordinates, and second, why are they oriented horizontally and vertically?

The theory of Optimal Unsupervised Motor Learning described below provides a model which attempts to answer both questions. It automatically develops orthogonal internal coordinates since such coordinates can be used to minimize redundancy in the internal representation and simplify computation of motor commands. The selection of the internal coordinates will be based on the statistics of the components of the sensory data which are controllable, so that if horizontal and vertical movements are distinguished in the environment then these components will determine the orientation of intermediate channels. We can hypothesize that the horizontal and vertical directions are distinguished in the owl by their relation to sensory information generated from physical properties of the environment such as gravity or symmetry properties of the owl's head. In the simulation below, I show that reasonable assumptions on such symmetry properties are sufficient to guarantee horizontal and vertical orientations of the intermediate coordinate system.

## 3    OPTIMAL UNSUPERVISED MOTOR LEARNING

Optimal Unsupervised Motor Learning (OUML) attempts to invert the dynamics of an unknown plant while maintaining control of the most important modes (Sanger 1993). Figure 1 shows the general structure of the control loop, where the plant $P$ maps motor commands $u$ into sensory outputs $y = Pu$, the adaptive sensory transformation $G$ maps sensory data $y$ into a reduced order intermediate representation $z = Gy$, and the adaptive motor transformation $N$ maps desired values of $z$ into the motor commands $u = Nz$ which achieve them. Let $\hat{z} = GPNz$ be the value of the intermediate variables after movement, and $\hat{y} = PNGy$ be the resulting value of the sensory variables. For any chosen value of $z$ we want $\hat{z} = z$, so that we successfully control the intermediate variables.

In (Sanger 1993) it was proposed that we want to choose $z$ to have lower dimensionality than $y$ and to represent only the coordinates which are most important for controlling the desired behavior. Thus, in general, $\hat{y} \neq y$ and $\|y - \hat{y}\|$ is the performance error. OUML can then be described as

1. Minimize the movement error $\|\hat{y} - y\|$
2. Subject to accurate control $\hat{z} = z$.

These criteria lead to a choice of internal representation that maximizes the loop gain through the plant.

**Theorem 1:**    (Sanger 1993) *For any sensory mapping $G$ there exists a motor mapping $N$ such that $\hat{z} = z$, and $\mathcal{E} = E[\|y - \hat{y}\|]$ is minimized when $G$ is chosen to minimize $E[\|y - \bar{G}^{-1}Gy\|]$, where $\bar{G}^{-1}$ is such that $G\bar{G}^{-1} = I$.*

The function $\bar{G}$ is an arbitrary right inverse of $G$, and this function determines the asymptotic values of the unobserved modes. In other words, since $G$ in general is dimensionality-reducing, $z = Gy$ will not respond to all the modes in $y$ so that dissimilar states may project to identical intermediate control variables $z$. The

| Plant$^{-1}$ | Motor | Sensory |
|---|---|---|
| Linear | Linear | Eigenvectors of $E[yy^T]$ |
| RBF | Linear | Eigenvectors of basis function outputs |
| Polynomial | Polynomial | Eigenvectors of basis function outputs |

Figure 2: Special cases of Theorem 1. If the plant inverse is linear or can be approximated using a sum of radial basis functions or a polynomial, then simple closed-form solutions exist for the optimal sensory network and the motor network only needs to be linear or polynomial.

function $\bar{G}^{-1}G$ is a projection operator that determines the resulting plant output $\hat{y}$ for any desired value of $y$. Unsupervised motor learning is "optimal" when the projection surface determined by $\bar{G}^{-1}G$ is the best approximation to the statistical density of desired values of $y$.

Without detailed knowledge of the plant, it may be difficult to find the general solution described by the theorem. Fortunately, there are several important special cases in which simple closed-form solutions exist. These cases are summarized in figure 2 and are determined by the class of functions to which the plant inverse belongs. If the plant inverse can be approximated as a sum of radial basis functions, then the motor network need only be linear and the optimal sensory network is given by the eigenvectors of the autocorrelation matrix of the basis function outputs (as in (Sanger 1991a)). If the plant inverse can be approximated as a polynomial over a set of basis functions (as in (Sanger 1991b)), then the motor network needs to be a polynomial, and again the optimal sensory network is given by the eigenvectors of the autocorrelation matrix of the basis function outputs.

Since the model of the barn owl proposed below has a linear inverse we are interested in the linear case, so we know that the mappings $N$ and $G$ need only be linear and that the optimal value of $G$ is given by the eigenvectors of the autocorrelation matrix of the plant outputs $y$. In fact, it can be shown that the optimal $N$ and $G$ are given by the matrices of left and right singular vectors of the plant inverse (Sanger 1993).

Although several algorithms for iterative computation of eigenvectors exist, until recently there were no iterative algorithms for finding the left and right singular vectors. I have developed two such algorithms, called the "Double Generalized Hebbian Algorithm" (DGHA) and the "Orthogonal Asymmetric Encoder" (OAE). (These algorithms are described in detail elsewhere in this volume.) DGHA is described by:

$$\Delta G = \gamma(zy^T - \text{LT}[zz^T]G)$$
$$\Delta N^T = \gamma(zu^T - \text{LT}[zz^T]N^T)$$

while OAE is described by:

$$\Delta G = \gamma(\hat{z}y^T - \text{LT}[\hat{z}\hat{z}^T]G)$$
$$\Delta N^T = \gamma(Gy - \text{LT}[GG^T]\hat{z})u^T$$

where LT[ ] is an operator that sets the above diagonal elements of its matrix argument to zero, $y = Pu$, $z = Gy$, $\hat{z} = N^Tu$, and $\gamma$ is a learning rate constant.

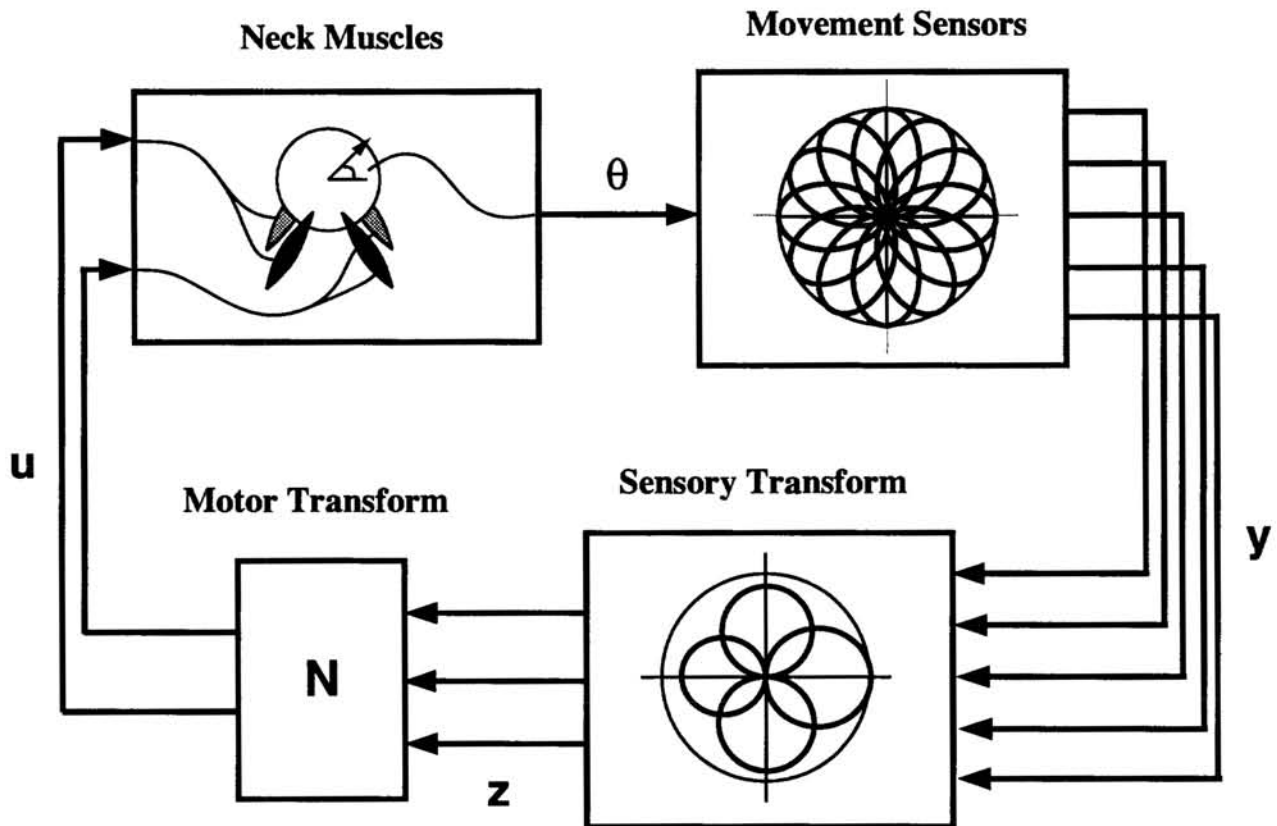

Figure 3: Owl model, and simulation results. The "Sensory Transform" box shows the orientation tuning of the learned internal representation.

## 4   SIMULATION

I use OUML to simulate the owl head movement experiments described in (Masino and Knudsen 1990), and I predict the form of the internal motor representation. I assume a simple model for the owl head using two sets of muscles which are not aligned with either the horizontal or the vertical direction (see the upper left block of figure 3). This model is an extreme oversimplification of the large number of muscle groups present in the barn owl neck, but it will serve to illustrate the case of muscles which do not distinguish the horizontal and vertical directions.

I assume that during learning the owl gives essentially random commands to the muscles, but that the physics of head movement result in a slight predominance of either vertical or horizontal motion. This assumption comes from the symmetry properties of the owl head, for which it is reasonable to expect that the axes of rotational symmetry lie in the coronal, sagittal, and transverse planes, and that the moments of inertia about these axes are not equal. I model sensory receptors using a set of 12 oriented directionally-tuned units, each with a half-bandwidth at half-height of 15 degrees (see the upper right block of figure 3). Together, the Neck Muscles and Movement Sensors (the two upper blocks of figure 3) form the model of the plant which transforms motor commands $u$ into sensory outputs $y$. Although this plant is nonlinear, it can be shown to have an approximately linear inverse on

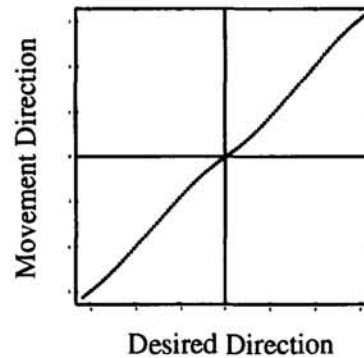

Figure 4: Unsupervised Motor Learning successfully controls the owl head simulation.

its range.

The sensory units are connected through an adaptive linear network $G$ to three intermediate units which will become the internal coordinate system $z$. The three intermediate units are then connected back to the motor outputs through a motor network $N$ so that desired sensory states can be mapped onto the motor commands necessary to produce them. The sensory to intermediate and intermediate to motor mappings were allowed to adapt to 1000 random head movements, with learning controlled by DGHA.

## 5   RESULTS

After learning, the first intermediate unit responded to the existence of a motion, and did not indicate its direction. The second and third units became broadly tuned to orthogonal directions. Over many repeated learning sessions starting from random initial conditions, it was found that the intermediate units were always aligned with the horizontal and vertical axes and never with the diagonal motor axes. The resulting orientation tuning from a typical session is shown in the lower right box of figure 3.

Note that these units are much more broadly tuned than the movement sensors (the half-bandwidth at half-height is 45 degrees). The orientation of the internal channels is determined by the assumed symmetry properties of the owl head. This information is available to the owl as sensory data, and OUML allows it to determine the motor representation. The system has successfully inverted the plant, as shown in figure 4.

(Masino and Knudsen 1990) investigated the intermediate representations in the owl by taking advantage of the refractory period of the internal channels. It was found that if two electrical stimuli which at long latency tended to move the owl's head in directions located in adjacent quadrants were instead presented at short latency, the second head movement would be aligned with either the horizontal or vertical axis. Figure 5 shows the general form of the experimental results, which are consistent with the hypothesis that there are four independent channels coding

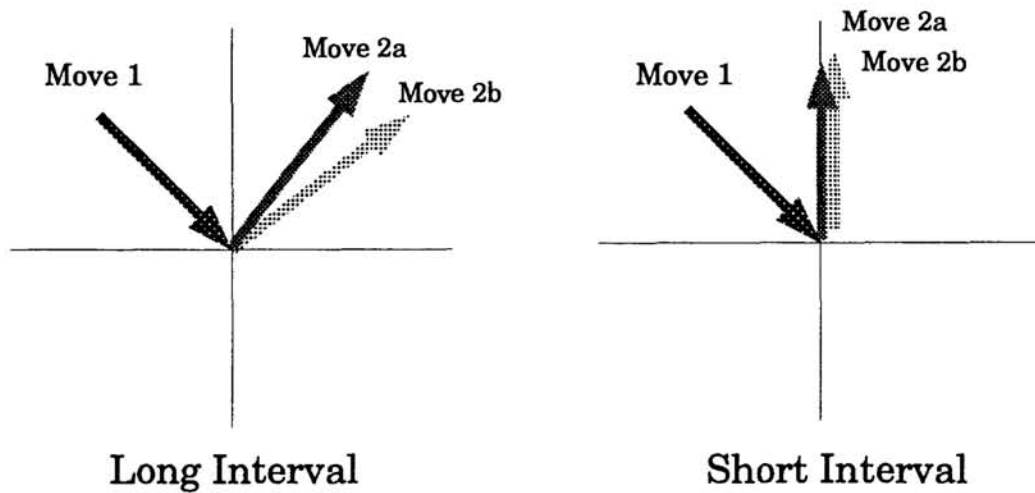

Figure 5: Schematic description of the owl head movement experiment. At long interstimulus intervals (ISI), moves 2a and 2b move up and to the right, but at short ISI the rightward channel is refractory from move 1 and thus moves 2a and 2b only have an upward component.

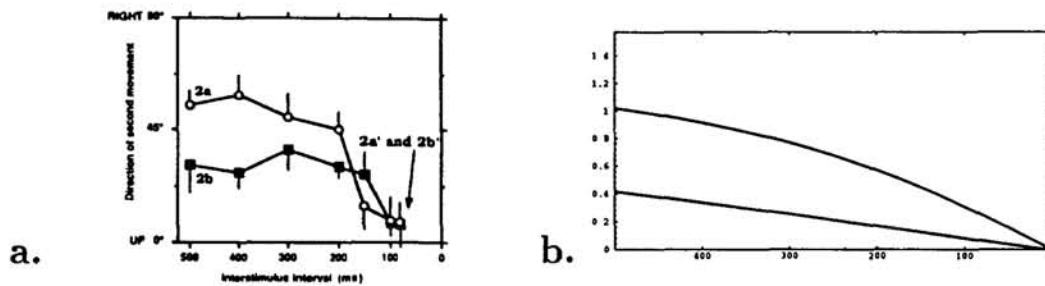

Figure 6: Movements align with the vertical axis as the ISI shortens. **a.** Owl data (reprinted with permission from (Masino and Knudsen 1990)). **b.** Simulation results.

the direction of head movement, and that the first movement makes either the left, right, up, or down channels refractory. As the interstimulus interval (ISI) is shortened, the alignment of the second movement with the horizontal or vertical axis becomes more pronounced. This is shown in figure 6a for the barn owl and 6b for the simulation. If we stimulate sites that move in many different directions, we find that at short latency the second movement always aligns with the horizontal or vertical axis, as shown in figure 7a for the owl and figure 7b for the simulation.

# 6    CONCLUSION

Optimal Unsupervised Motor Learning provides a model for adaptation in low-level motor systems. It predicts the development of orthogonal intermediate representations whose orientation is determined by the statistics of the controllable components of the sensory environment. The existence of iterative neural algorithms for both linear and nonlinear plants allows simulation of biological systems, and I have

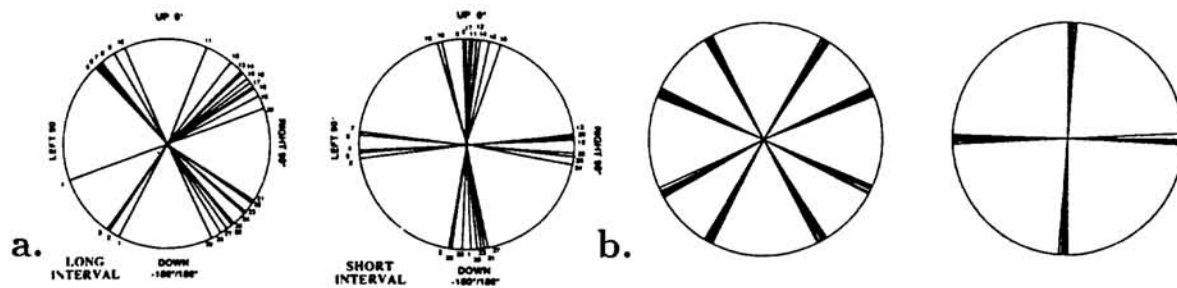

Figure 7: At long ISI, the second movement can occur in many directions, but at short ISI will tend to align with the horizontal or vertical axis. **a.** Owl data (reprinted with permission from (Masino and Knudsen 1990)). **b.** Simulation results.

shown that the optimal internal representation predicts the horizontal and vertical alignment of the internal channels for barn owl head movement.

## Acknowledgements

Thanks are due to Tom Masino for helpful discussions as well as for allowing reproduction of the figures from (Masino and Knudsen 1990). This report describes research done within the laboratory of Dr. Emilio Bizzi in the department of Brain and Cognitive Sciences at MIT. The author was supported during this work by a National Defense Science and Engineering Graduate Fellowship, and by NIH grants 5R37AR26710 and 5R01NS09343 to Dr. Bizzi.

## References

Becker S., 1992, *An Information-Theoretic Unsupervised Learning Algorithm for Neural Networks*, PhD thesis, Univ. Toronto Dept. Computer Science.

Kohonen T., 1982, Self-organized formation of topologically correct feature maps, *Biological Cybernetics*, 43:59–69.

Linsker R., 1989, How to generate ordered maps by maximizing the mutual information between input and output signals, *Neural Computation*, 1:402–411.

Masino T., Knudsen E. I., 1990, Horizontal and vertical components of head movement are controlled by distinct neural circuits in the barn owl, *Nature*, 345:434–437.

Sanger T. D., 1989, Optimal unsupervised learning in a single-layer linear feedforward neural network, *Neural Networks*, 2:459–473.

Sanger T. D., 1991a, Optimal hidden units for two-layer nonlinear feedforward neural networks, *International Journal of Pattern Recognition and Artificial Intelligence*, 5(4):545–561, Also appears in C. H. Chen, ed., *Neural Networks in Pattern Recognition and Their Applications*, World Scientific, 1991, pp. 43-59.

Sanger T. D., 1991b, A tree-structured adaptive network for function approximation in high dimensional spaces, *IEEE Trans. Neural Networks*, 2(2):285–293.

Sanger T. D., 1993, Optimal unsupervised motor learning, *IEEE Trans. Neural Networks*, in press.